# Batch and On-line Parameter Estimation of Gaussian Mixtures Based on the Joint Entropy

**Yoram Singer**
AT&T Labs
singer@research.att.com

**Manfred K. Warmuth**
University of California, Santa Cruz
manfred@cse.ucsc.edu

## Abstract

We describe a new iterative method for parameter estimation of Gaussian mixtures. The new method is based on a framework developed by Kivinen and Warmuth for supervised on-line learning. In contrast to gradient descent and EM, which estimate the mixture's covariance matrices, the proposed method estimates the *inverses* of the covariance matrices. Furthermore, the new parameter estimation procedure can be applied in both on-line and batch settings. We show experimentally that it is typically faster than EM, and usually requires about half as many iterations as EM.

## 1 Introduction

Mixture models, in particular mixtures of Gaussians, have been a popular tool for density estimation, clustering, and un-supervised learning with a wide range of applications (see for instance [5, 2] and the references therein). Mixture models are one of the most useful tools for handling incomplete data, in particular hidden variables. For Gaussian mixtures the hidden variables indicate for each data point the index of the Gaussian that generated it. Thus, the model is specified by a joint density between the observed and hidden variables. The common technique used for estimating the parameters of a stochastic source with hidden variables is the EM algorithm. In this paper we describe a new technique for estimating the parameters of Gaussian mixtures. The new parameter estimation method is based on a framework developed by Kivinen and Warmuth [8] for supervised on-line learning. This framework was successfully used in a large number of supervised and un-supervised problems (see for instance [7, 6, 9, 1]).

Our goal is to find a local minimum of a loss function which, in our case, is the negative log likelihood induced by a mixture of Gaussians. However, rather than minimizing the

loss directly we add a term measuring the distance of the new parameters to the old ones. This distance is useful for iterative parameter estimation procedures. Its purpose is to keep the new parameters close to the old ones. The method for deriving iterative parameter estimation can be used in batch settings as well as on-line settings where the parameters are updated after each observation. The distance used for deriving the parameter estimation method in this paper is the relative entropy between the old and new joint density of the observed and hidden variables. For brevity we term the new iterative parameter estimation method the *joint-entropy* (JE) update.

The JE update shares a common characteristic with the Expectation Maximization [4, 10] algorithm as it first calculates the same expectations. However, it replaces the maximization step with a different update of the parameters. For instance, it updates the inverse of the covariance matrix of each Gaussian in the mixture, rather than the covariance matrices themselves. We found in our experiments that the JE update often requires half as many iterations as EM. It is also straightforward to modify the proposed parameter estimation method for on-line setting where the parameters are updated after each new observation. As we demonstrate in our experiments with digit recognition, the on-line version of the JE update is especially useful in situations where the observations are generated by a non-stationary stochastic source.

## 2 Notation and preliminaries

Let $S$ be a sequence of training examples $\langle x_1, x_2, \ldots, x_N \rangle$ where each $x_i$ is a $d$-dimensional vector in $\mathbb{R}^d$. To model the distribution of the examples we use $m$ $d$-dimensional Gaussians. The parameters of the $i$-th Gaussian are denoted by $\Theta_i$ and they include the mean-vector and the covariance matrix

$$\mu_i = E(\mathbf{x}|\Theta_i) \quad C_i = E((\mathbf{x} - \mu_i)(\mathbf{x} - \mu_i)^T |\Theta_i) .$$

The density function of the $i$th Gaussian, denoted $P(\mathbf{x}|\Theta_i)$, is

$$P(\mathbf{x}|\Theta_i) = (2\pi)^{-d/2}|\mathbf{C}_i|^{-1/2}e^{-\frac{1}{2}(\mathbf{x}-\mu_i)^T \mathbf{C}_i^{-1}(\mathbf{x}-\mu_i)} .$$

We denote the entire set of parameters of a Gaussian mixture by $\Theta = \{\Theta_i\}_{i=1}^m = \{w_i, \mu_i, C_i\}_{i=1}^m$ where $\mathbf{w} = (w_1, \ldots, w_m)$ is a non-negative vector of mixture coefficients such that $\sum_{i=1}^m w_i = 1$. We denote by $P(\mathbf{x}|\Theta) = \sum_{i=1}^m w_i P(\mathbf{x}|\Theta_i)$ the likelihood of an observation $\mathbf{x}$ according to a Gaussian mixture with parameters $\Theta$. Let $\Theta_i$ and $\tilde{\Theta}_i$ be two Gaussian distributions. For brevity, we denote by $E_i(Z)$ and $\tilde{E}_i(Z)$ the expectation of a random variable $Z$ with respect to $\Theta_i$ and $\tilde{\Theta}_i$. Let $f$ be a parametric function whose parameters constitute a matrix $A = (a_{ij})$. We denote by $\partial f / \partial A$ the matrix of partial derivatives of $f$ with respect to the elements in $A$. That is, the $ij$ element of $\partial f / \partial A$ is $\partial f / \partial a_{ij}$. Similarly, let $B = (b_{ij}(x))$ a matrix whose elements are functions of a scalar $x$. Then, we denote by $dB/dx$ the matrix of derivatives of the elements in $B$ with respect to $x$, namely, the $ij$ element of $dB/dx$ is $db_{ij}(x)/dx$.

## 3 The framework for deriving updates

Kivinen and Warmuth [8] introduced a general framework for deriving on-line parameter updates. In this section we describe how to apply their framework for the problem of

parameter estimation of Gaussian mixtures in a batch setting. We later discuss how a simple modification gives the on-line updates.

Given a set of data points $S$ in $\mathbb{R}^d$ and a number $m$, the goal is to find a set of $m$ Gaussians that minimize the loss on the data, denoted as $\text{loss}(S|\Theta)$. For density estimation the natural loss function is the negative log-likelihood of the data $\text{loss}(S|\Theta) = -(1/|S|) \ln P(S|\Theta) \overset{\text{def}}{=} -(1/|S|) \sum_{\mathbf{x} \in S} \ln P(\mathbf{x}|\Theta)$. The best parameters which minimize the above loss cannot be found analytically. The common approach is to use iterative methods such as EM [4, 10] to find a local minimizer of the loss.

In an iterative parameter estimation framework we are given the old set of parameters $\Theta^t$ and we need to find a set of new parameters $\Theta^{t+1}$ that induce smaller loss. The framework introduced by Kivinen and Warmuth [8] deviates from the common approaches as it also requires to the new parameter vector to stay "close" to the old set of parameters which incorporates all that was learned in the previous iterations. The distance of the new parameter setting $\Theta^{t+1}$ from the old setting $\Theta^t$ is measured by a non-negative distance function $\Delta(\Theta^{t+1}, \Theta^t)$. We now search for a new set of parameters $\Theta^{t+1}$ that minimizes the distance summed with the loss multiplied by $\eta$. Here $\eta$ is a non-negative number measuring the relative importance of the distance versus the loss. This parameter $\eta$ will become the learning rate of the update. More formally, the update is found by setting $\Theta^{t+1} = \arg\min_{\widetilde{\Theta}} U^t(\widetilde{\Theta})$ where $U^t(\widetilde{\Theta}) = \Delta(\widetilde{\Theta}, \Theta^t) + \eta \, \text{loss}(S|\widetilde{\Theta}) + \lambda(\sum_{i=1}^m \widetilde{w}_i - 1)$. (We use a Lagrange multiplier $\lambda$ to enforce the constraint that the mixture coefficients sum to one.) By choosing the apropriate distance function and $\eta = 1$ one can show that EM becomes the above update.

For most distance functions and learning rates the minimizer of the function $U^t(\widetilde{\Theta})$ cannot be found analytically as both the distance function and the log-likelihood are usually non-linear in $\widetilde{\Theta}$. Instead, we expand the log-likelihood using a first order Taylor expansion around the old parameter setting. This approximation degrades the further the new parameter values are from the old ones, which further motivates the use of the distance function $\Delta(\widetilde{\Theta}, \Theta^t)$ (see also the discussion in [7]). We now seek a new set of parameters $\Theta^{t+1} = \arg\min_{\widetilde{\Theta}} V^t(\widetilde{\Theta})$ where

$$V^t(\widetilde{\Theta}) = \Delta(\widetilde{\Theta}, \Theta^t) + \eta \left( \text{loss}(S|\Theta^t) + (\widetilde{\Theta} - \Theta^t) \cdot \nabla_\Theta \text{loss}(S|\Theta^t) \right) + \lambda(\sum_{i=1}^m \widetilde{w}_i - 1) . \quad (1)$$

Here $\nabla_\Theta \text{loss}(S|\Theta^t)$ denotes the gradient of the loss at $\Theta^t$. We use the above method Eq. (1) to derive the updates of this paper. For density estimation, it is natural to use the relative entropy between the new and old density as a distance. In this paper we use the joint density between the observed (data points) and hidden variables (the indices of the Gaussians). This motivates the name joint-entropy update.

## 4   Entropy based distance functions

We first consider the relative entropy between the new and old parameter parameters of a *single* Gaussian. Using the notation introduced in Sec. 2, the relative entropy between two Gaussian distributions denoted by $\widetilde{\Theta}_i, \Theta_i$ is

$$\Delta(\widetilde{\Theta}_i, \Theta_i) \overset{\text{def}}{=} \int_{\mathbf{x} \in \mathbb{R}^d} P(\mathbf{x}|\widetilde{\Theta}_i) \ln \frac{P(\mathbf{x}|\widetilde{\Theta}_i)}{P(\mathbf{x}|\Theta_i)} \, d\mathbf{x}$$

$$= \frac{1}{2} \ln \frac{|\mathbf{C}_i|}{|\widetilde{\mathbf{C}}_i|} - \frac{1}{2}\widetilde{E}_i \left( (\mathbf{x} - \widetilde{\mu}_i)^T \widetilde{\mathbf{C}}_i^{-1}(\mathbf{x} - \widetilde{\mu}_i) \right) + \frac{1}{2}\widetilde{E}_i \left( (\mathbf{x} - \mu_i)^T \mathbf{C}_i^{-1}(\mathbf{x} - \mu_i) \right)$$

Using standard (though tedious) algebra we can rewrite the expectations as follows:

$$\Delta(\widetilde{\Theta}_i, \Theta_i) = \tfrac{1}{2}\ln\frac{|\mathbf{C}_i|}{|\widetilde{\mathbf{C}}_i|} - \frac{d}{2} + \tfrac{1}{2}\mathrm{tr}(\mathbf{C}_i^{-1}\widetilde{\mathbf{C}}_i) + \tfrac{1}{2}(\widetilde{\mu}_i - \mu)^T\mathbf{C}_i^{-1}(\widetilde{\mu}_i - \mu_i)\,. \qquad (2)$$

The relative entropy between the new and the old *mixture* models is the following

$$\Delta(\widetilde{\Theta}, \Theta) \stackrel{\mathrm{def}}{=} \int_X P(\mathbf{x}|\widetilde{\Theta})\ln\frac{P(\mathbf{x}|\widetilde{\Theta})}{P(\mathbf{x}|\Theta)}d\mathbf{x} = \int_X \sum_{i=1}^m \widetilde{w}_i P(\mathbf{x}|\widetilde{\Theta}_i)\ln\frac{\sum_{i=1}^m \widetilde{w}_i P(\mathbf{x}|\widetilde{\Theta}_i)}{\sum_{i=1}^m w_i P(\mathbf{x}|\Theta_i)}d\mathbf{x}\,. \qquad (3)$$

Ideally, we would like to use the above distance function in $V^t$ to give us an update of $\widetilde{\Theta}$ in terms of $\Theta$. However, there isn't a closed form expression for Eq. (3). Although the relative entropy between two Gaussians is a convex function in their parameters, the relative entropy between two Gaussian mixtures is non-convex. Thus, the loss function $V^t(\widetilde{\Theta})$ may have multiple minima, making the problem of finding $\arg\min_{\widetilde{\Theta}} V^t(\widetilde{\Theta})$ difficult.

In order to sidestep this problem we use the log-sum inequality [3] to obtain an upper bound for the distance function $\Delta(\widetilde{\Theta}, \Theta)$. We denote this upper bound as $\widehat{\Delta}(\widetilde{\Theta}, \Theta)$.

$$\Delta(\widetilde{\Theta}, \Theta) \leq \widehat{\Delta}(\widetilde{\Theta}, \Theta) \stackrel{\mathrm{def}}{=} \int_{\mathbf{x}} \sum_{i=1}^m (\mathbf{x}, i|\widetilde{\Theta})\ln\frac{P(\mathbf{x}, i|\widetilde{\Theta})}{P(\mathbf{x}, i|\Theta)}d\mathbf{x} = \int_{\mathbf{x}} \sum_{i=1}^m \widetilde{w}_i P(\mathbf{x}|\widetilde{\Theta}_i)\ln\frac{\widetilde{w}_i P(\mathbf{x}|\widetilde{\Theta}_i)}{w_i P(\mathbf{x}|\Theta_i)}d\mathbf{x}$$

$$= \sum_{i=1}^m \widetilde{w}_i \ln\frac{\widetilde{w}_i}{w_i} + \sum_{i=1}^m \widetilde{w}_i \int_{\mathbf{x}} P(\mathbf{x}|\widetilde{\Theta}_i)\ln\frac{P(\mathbf{x}|\widetilde{\Theta}_i)}{P(\mathbf{x}|\Theta_i)}d\mathbf{x} = \sum_{i=1}^m \widetilde{w}_i \ln\frac{\widetilde{w}_i}{w_i} + \sum_{i=1}^m \widetilde{w}_i \Delta(\widetilde{\Theta}_i, \Theta_i)\,. \qquad (4)$$

We call the new distance function $\widehat{\Delta}(\widetilde{\Theta}, \Theta)$ the *joint-entropy distance*. Note that in this distance the parameters of $\widetilde{w}_i$ and $w_i$ are "coupled" in the sense that it is a convex combination of the distances $\Delta(\widetilde{\Theta}_i, \Theta_i)$. In particular, $\widehat{\Delta}(\widetilde{\Theta}, \Theta)$ as a function of the parameters $\widetilde{w}_i$, $\widetilde{\mu}_i$, $\widetilde{\mathbf{C}}_i$ does not remain constant any more when the parameters of the individual Gaussians are permuted. Furthermore, $\widehat{\Delta}(\widetilde{\Theta}, \Theta)$ is also is sufficiently convex so that finding the minimizer of $V^t$ is possible (see below).

## 5 The updates

We are now ready to derive the new parameter estimation scheme. This is done by setting the partial derivatives of $V^t$, with respect to $\widetilde{\Theta}$, to 0. That is, our problem consists of solving the following equations

$$\frac{\partial\widehat{\Delta}(\widetilde{\Theta}, \Theta)}{\partial\widetilde{w}_i} - \frac{\eta}{|S|}\frac{\partial\ln P(S|\Theta)}{\partial w_i} + \lambda = 0, \quad \frac{\partial\widehat{\Delta}(\widetilde{\Theta}, \Theta)}{\partial\widetilde{\mu}_i} - \frac{\eta}{|S|}\frac{\partial\ln P(S|\Theta)}{\partial\mu_i} = 0, \quad \frac{\partial\widehat{\Delta}(\widetilde{\Theta}, \Theta)}{\partial\widetilde{\mathbf{C}}_i} - \frac{\eta}{|S|}\frac{\partial\ln P(S|\Theta)}{\partial\mathbf{C}_i} = 0.$$

We now use the fact that $\mathbf{C}_i$ and thus $\mathbf{C}_i^{-1}$ is symmetric. The derivatives of $\widehat{\Delta}(\widetilde{\Theta}, \Theta)$, as defined by Eq. (4) and Eq. (2), with respect to $\widetilde{w}_i, \widetilde{\mu}_i$ and $\widetilde{\mathbf{C}}_i$, are

$$\frac{\partial\widehat{\Delta}(\widetilde{\Theta}, \Theta)}{\partial\widetilde{w}_i} = \ln\frac{\widetilde{w}_i}{w_i} + 1 + \tfrac{1}{2}\ln\frac{|\mathbf{C}_i|}{|\widetilde{\mathbf{C}}_i|} - \frac{d}{2} + \tfrac{1}{2}\mathrm{tr}(\mathbf{C}_i^{-1}\widetilde{\mathbf{C}}_i) + \tfrac{1}{2}(\widetilde{\mu}_i - \mu)^T\mathbf{C}_i^{-1}(\widetilde{\mu}_i - \mu_i) \qquad (5)$$

$$\frac{\partial\widehat{\Delta}(\widetilde{\Theta}, \Theta)}{\partial\widetilde{\mu}_i} = \widetilde{w}_i\mathbf{C}_i^{-1}(\widetilde{\mu}_i - \mu_i) \qquad (6)$$

$$\frac{\partial\widehat{\Delta}(\widetilde{\Theta}, \Theta)}{\partial\widetilde{\mathbf{C}}_i} = \tfrac{1}{2}\widetilde{w}_i(-\widetilde{\mathbf{C}}_i^{-1} + \mathbf{C}_i^{-1})\,. \qquad (7)$$

To simplify the notation throughout the rest of the paper we define the following variables

$$\beta_i(\mathbf{x}) \overset{\text{def}}{=} \frac{P(\mathbf{x}|\Theta_i)}{P(\mathbf{x}|\Theta)} \quad \text{and} \quad \alpha_i(\mathbf{x}) \overset{\text{def}}{=} \frac{w_i P(\mathbf{x}|\Theta_i)}{P(\mathbf{x}|\Theta)} = P(i|\mathbf{x}, \Theta_i) = w_i \beta_i(\mathbf{x}).$$

The partial derivatives of the log-likelihood are computed similarly:

$$\frac{\partial \ln P(S|\Theta)}{\partial w_i} = \sum_{\mathbf{x} \in S} \frac{P(\mathbf{x}|\Theta_i)}{P(\mathbf{x}|\Theta)} = \sum_{\mathbf{x} \in S} \beta_i(\mathbf{x}) \tag{8}$$

$$\frac{\partial \ln P(S|\Theta)}{\partial \mu_i} = \sum_{\mathbf{x} \in S} \frac{w_i P(\mathbf{x}|\Theta_i)}{P(\mathbf{x}|\Theta)} \mathbf{C}_i^{-1}(\mathbf{x} - \mu_i) = \sum_{\mathbf{x} \in S} \alpha_i(\mathbf{x}) \, \mathbf{C}_i^{-1}(\mathbf{x} - \mu_i) \tag{9}$$

$$\frac{\partial \ln P(S|\Theta)}{\partial \mathbf{C}_i} = -\frac{1}{2} \sum_{\mathbf{x} \in S} \frac{w_i P(\mathbf{x}|\Theta_i)}{P(\mathbf{x}|\Theta)} (\mathbf{C}_i^{-1} - \mathbf{C}_i^{-1}(\mathbf{x} - \mu_i)(\mathbf{x} - \mu_i)^T \mathbf{C}_i^{-1})$$

$$= -\frac{1}{2} \sum_{\mathbf{x} \in S} \alpha_i(\mathbf{x}) (\mathbf{C}_i^{-1} - \mathbf{C}_i^{-1}(\mathbf{x} - \mu_i)(\mathbf{x} - \mu_i)^T \mathbf{C}_i^{-1}). \tag{10}$$

We now need to decide on an order for updating the parameter classes $w_i$, $\mu_i$, and $\mathbf{C}_i$. We use the same order that EM uses, namely, $w_i$, then $\mu_i$, and finally, $\mathbf{C}_i$. (After doing one pass over all three groups we start again using the same order.) Using this order results in a simplified set of equations as several terms in Eq. (5) cancel out. Denote the size of the sample by $N = |S|$. We now need to sum the derivatives from Eq. (5) and Eq. (8) while using the fact that the Lagrange multiplier $\lambda$ simply assures that the new weight $\tilde{w}_i$ sum to one. By setting the result to zero, we get that

$$w_i \leftarrow \frac{w_i \exp\left(-\frac{\eta}{N} \sum_{\mathbf{x} \in S} \beta_i(\mathbf{x})\right)}{\sum_{j=1}^m w_j \exp\left(-\frac{\eta}{N} \sum_{\mathbf{x} \in S} \beta_i(\mathbf{x})\right)}. \tag{11}$$

Similarly, we sum Eq. (6) and Eq. (9), set the result to zero, and get that

$$\mu_i \leftarrow \mu_i + \frac{\eta}{N} \sum_{\mathbf{x} \in S} \beta_i(\mathbf{x}) (\mathbf{x} - \mu_i). \tag{12}$$

Finally, we do the same for $\mathbf{C}_i$. We sum Eq. (7) and Eq. (10) using the newly obtained $\mu_i$,

$$\mathbf{C}_i^{-1} \leftarrow \mathbf{C}_i^{-1} + \frac{\eta}{N} \sum_{\mathbf{x} \in S} \beta_i(\mathbf{x}) (\mathbf{C}_i^{-1} - \mathbf{C}_i^{-1}(\mathbf{x} - \mu_i)(\mathbf{x} - \mu_i)^T \mathbf{C}_i^{-1}). \tag{13}$$

We call the new iterative parameter estimation procedure the joint-entropy (JE) update. To summarize, the JE update is composed of the following alternating steps: We first calculate for each observation $\mathbf{x}$ the value $\beta_i(\mathbf{x}) = P(\mathbf{x}|\Theta_i)/P(\mathbf{x}|\Theta)$ and then update the parameters as given by Eq. (11), Eq. (12), and Eq. (13). The JE update and EM differ in several aspects. First, EM uses a simple update for the mixture weights $\mathbf{w}$. Second, EM uses the expectations (with respect to the current parameters) of the sufficient statistics [4] for $\mu_i$ and $C_i$ to find new sets of mean vectors and covariance matrices. The JE uses a (slightly different) weighted average of the observation and, in addition, it adds the old parameters. The learning rate $\eta$ determines the proportion to be used in summing the old parameters and the newly estimated parameters. Last, EM estimates the covariance matrices $\mathbf{C}_i$ whereas the new update estimates the *inverses*, $\mathbf{C}_i^{-1}$, of these matrices. Thus, it is potentially be more stable numerically in cases where the covariance matrices have small condition number.

To obtain an on-line procedure we need to update the parameters after each new observation at a time. That is, rather than summing over all $\mathbf{x} \in S$, for a new observation $\mathbf{x}_t$, we update

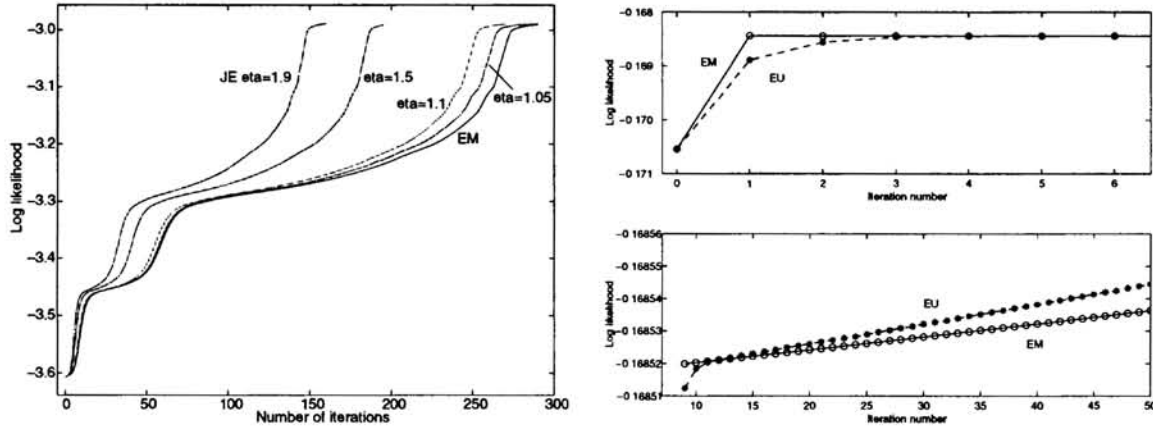

Figure 1: Left: comparison of the convergence rate of EM and the JE update with different learning rates. Right: example of a case where EM *initially* increases the likelihood faster than the JE update.

the parameters and get a new set of parameters $\Theta^{t+1}$ using the current parameters $\Theta^t$. The new parameters are then used for inducing the likelihood of the next observation $\mathbf{x}_{t+1}$. The on-line parameter estimation procedure is composed of the following steps:

1. Set: $\beta_i(\mathbf{x}_t) = \frac{P(\mathbf{x}_t|\Theta_i)}{P(\mathbf{x}_t|\Theta)}$.
2. Parameter updates:
   (a) $w_i \leftarrow w_i \exp\left(-\eta_t \beta_i(\mathbf{x}_t)\right) / \sum_{j=1}^m w_j \exp\left(-\eta_t \beta_i(\mathbf{x}_t)\right)$
   (b) $\mu_i \leftarrow \mu_i + \eta_t \, \beta_i(\mathbf{x}_t) \, (\mathbf{x}_t - \mu_i)$
   (c) $\mathbf{C}_i^{-1} \leftarrow \mathbf{C}_i^{-1} + \eta_t \, \beta_i(\mathbf{x}_t) \left(\mathbf{C}_i^{-1} - \mathbf{C}_i^{-1}(\mathbf{x}_t - \mu_i)(\mathbf{x}_t - \mu_i)^T \mathbf{C}_i^{-1}\right)$.

To guarantee convergence of the on-line update one should use a diminishing learning rate, that is $\eta_t \to 0$ as $t \to \infty$ (for further motivation see [11]).

## 6   Experiments

We conducted numerous experiments with the new update. Due to the lack of space we describe here only two. In the first experiment we compared the JE update and EM in batch settings. We generated data from Gaussian mixture distributions with varying number of components ($m = 2$ to $100$) and dimensions ($d = 2$ to $20$). Due to the lack of space we describe here results obtained from only one setting. In this setting the examples were generated by a mixture of 5 components with $\mathbf{w} = (0.4, 0.3, 0.2, 0.05, 0.05)$. The mean vectors were the 5 standard unit vectors in the Euclidean space $\mathbb{R}^5$ and we set all of covariances matrices to the identity matrix. We generated 1000 examples. We then run EM and the JE update with different learning rates ($\eta = 1.9, 1.5, 1.1, 1.05$). To make sure that all the runs will end in the same local maximum we fist performed three EM iterations. The results are shown on the left hand side of Figure 1. In this setting, the JE update with high learning rates achieves much faster convergence than EM. We would like to note that this behavior is by no means esoteric – most of our experiments data yielded similar results.

We found a different behavior in low dimensional settings. On the right hand side of Figure 1 we show convergence rate results for a mixture containing two components each of which is a single dimension Gaussians. The mean of the two components were located

at 1 and $-1$ with the same variance of 2. Thus, there is a significant "overlap" between the two Gaussian constituting the mixture. The mixture weight vector was $(0.5, 0.5)$. We generated 50 examples according to this distribution and initialized the parameters as follows: $\mu_1 = 0.01, \mu_2 = -0.01$, $\sigma_1 = \sigma_2 = 2$, $w_1 = w_2 = 0.5$ We see that initially EM increases the likelihood much faster than the JE update. Eventually, the JE update convergences faster than EM when using a small learning rate (in the example appearing in Figure 1 we set $\eta = 1.05$). However, in this setting, the JE update diverges when learning rates larger than $\eta = 1.1$ are used. This behavior underscores the advantages of both methods. EM uses a fixed learning rate and is guaranteed to converge to a local maximum of the likelihood, under conditions that typically hold for mixture of Gaussians [4, 12]. the JE update, on the other hand, encompasses a learning rate and in many settings it converges much faster than EM. However, the superior performance in high dimensional cases demands its price in low dimensional "dense" cases. Namely, a very conservative learning rate, which is hard to tune, need to be used. In these cases, EM is a better alternative, offering almost the same convergence rate without the need to tune any parameters.

**Acknowledgments**  Thanks to Duncan Herring for careful proof reading and providing us with interesting data sets.

# References

[1] E. Bauer, D. Koller, and Y. Singer. Update rules for parameter estimation in Bayesian networks. In *Proc. of the 13th Annual Conf. on Uncertainty in AI*, pages 3–13, 1997.

[2] C.M. Bishop. *Neural Networks and Pattern Recognition*. Oxford Univ. Press, 1995.

[3] Thomas M. Cover and Joy A. Thomas. *Elements of Information Theory*. Wiley, 1991.

[4] A.P. Dempster, N.M. Laird, and D.B. Rubin. Maximum-likelihood from incomplete data via the EM algorithm. *Journal of the Royal Statistical Society*, B39:1–38, 1977.

[5] R.O. Duda and P.E. Hart. *Pattern Classification and Scene Analysis*. Wiley, 1973.

[6] D. P. Helmbold, J. Kivinen, and M.K. Warmuth. Worst-case loss bounds for sigmoided neurons. In *Advances in Neural Information Processing Systems 7*, pages 309–315, 1995.

[7] D.P. Helmbold, R.E. Schapire, Y.Singer, and M.K. Warmuth. A comparison of new and old algorithms for a mixture estimation problem. *Machine Learning*, Vol. 7, 1997.

[8] J. Kivinen and M.K. Warmuth. Additive versus exponentiated gradient updates for linear prediction. *Information and Computation*, 132(1):1–64, January 1997.

[9] J. Kivinen and M.K. Warmuth. Relative loss bounds for multidimensional regression problems. In *Advances in Neural Information Processing Systems 10*, 1997.

[10] R.A. Redner and H.F. Walker. Mixture densities, maximum likelihood and the EM algorithm. *SIAM Review*, 26(2), 1984.

[11] D.M. Titterington, A.F.M. Smith, and U.E. Makov. *Statistical Analysis of Finite Mixture Distributions*. Wiley, 1985.

[12] C.F. Wu. On the convergence properties of the EM algorithm. *Annals of Stat.*, 11:95–103, 1983.